# Machine Learning Applied to Perception: Decision-Images for Gender Classification

**Felix A. Wichmann and Arnulf B. A. Graf**
Max Planck Institute for Biological Cybernetics
Tübingen, Germany
`felix.wichmann@tuebingen.mpg.de`

**Eero P. Simoncelli**
Howard Hughes Medical Institute
Center for Neural Science
New York University, USA

**Heinrich H. Bülthoff and Bernhard Schölkopf**
Max Planck Institute for Biological Cybernetics
Tübingen, Germany

## Abstract

We study gender discrimination of human faces using a combination of psychophysical classification and discrimination experiments together with methods from machine learning. We reduce the dimensionality of a set of face images using principal component analysis, and then train a set of linear classifiers on this reduced representation (linear support vector machines (SVMs), relevance vector machines (RVMs), Fisher linear discriminant (FLD), and prototype (prot) classifiers) using human classification data. Because we combine a linear preprocessor with linear classifiers, the entire system acts as a linear classifier, allowing us to visualise the *decision-image* corresponding to the normal vector of the separating hyperplanes (SH) of each classifier. We predict that the female-to-maleness transition along the normal vector for classifiers closely mimicking human classification (SVM and RVM [1]) should be faster than the transition along any other direction. A psychophysical discrimination experiment using the decision images as stimuli is consistent with this prediction.

## 1 Introduction

One of the central problems in vision science is to identify the features used by human subjects to classify visual stimuli. We combine machine learning and psychophysical techniques to gain insight into the algorithms used by human subjects during visual classification of faces. Comparing gender classification performance of humans to that of machines has attracted considerable attention in the past [2, 3, 4, 5]. The main novel aspect of our study is to analyse the machine algorithms to make inferences about the features used by human subjects, thus providing an alternative to psychophysical feature extraction techniques such as the "bubbles" [6] or the noise classification image [7] techniques. In this "machine-learning-psychophysics research" we first we train machine learning classifiers on the responses (labels) of human subjects to re-create the human decision boundaries by learning machines. Then we look for correlations between machine classifiers and sev-

eral characteristics of subjects' responses to the stimuli—proportion correct, reaction times (RT) and confidence ratings. Ideally this allows us to find preprocessor-classifier pairings that are closely aligned with the algorithm employed by the human brain for the task at hand. Thereafter we analyse properties of the machine closest to the human—in our case support vector machines (SVMs), and to slightly lesser degree, relevance vector machines (RVMs)—and make predictions about human behaviour based on machine properties.

In the current study we extract a *decision-image* containing the information relevant for classification by the machine classifiers. The decision-image $\vec{W}$ is the image corresponding to a vector $\vec{w}$ orthogonal to the SH of the classifier. The decision-image has the same dimensionality as the (input-) images—in our case $256 \times 256$—whereas the normal vector lives in the (reduced dimensionality) space after preprocessing—in our case in $200 \times 1$ after Principal Component Analysis (PCA). Second, we use $\vec{w}$ of the classifiers to generate novel stimuli by adding (or subtracting) various "amounts" ($\lambda\vec{w}$) to a genderless face in PCA space. The novel stimuli, images, $I(\lambda)$ are generated as $I(\lambda) = PCA^{-1}\lambda\frac{\vec{w}}{\|\vec{w}\|}$. We predict that the female-to-maleness transition along the vectors normal to the SHs, $\vec{w}_{\mathrm{SVM}}$ and $\vec{w}_{\mathrm{RVM}}$, should be significantly faster than those along the normal vectors of machine classifiers that do not correlate as well with human subjects. A psychophysical gender discrimination experiment confirms our predictions: the female-to-maleness axis of the SVM and, to a smaller extent, RVM, are more closely aligned with the human female-to-maleness axis than those of the prototype (Prot) and a Fisher linear discriminant (FLD) classifier.

## 2 Preprocessing and Machine Learning Methods

We preprocessed the faces using PCA. PCA is a good preprocessor in the current context since we have previously shown that in PCA-space strong correlations exist between man and machine [1]. Second, there is evidence that the PCA representation may be biologically-plausible [8]. The face stimuli were taken from the gender-balanced Max Planck Institute (MPI) face database[1] composed of 200 greyscale $256 \times 256$-pixel frontal views of human faces, yielding a data matrix $X \in \mathbb{R}^{200 \times 256^2}$. For the gender discrimination task we adhere to the following convention for the class labels: $y = -1$ for females and $y = +1$ for males. We consider no dimensionality reduction and keep all 200 components of the PCA. This implies that the reconstruction of the data from the PCA analysis is perfect and we can write: $E = \bar{X}B^T \Leftrightarrow \bar{X} = EB$ where $E \in \mathbb{R}^{200 \times 200}$ is the matrix of the encodings (each row is a PCA vector in the space of reduced dimensionality), $B \in \mathbb{R}^{200 \times 256^2}$ is the orthogonal basis matrix and $\bar{X}$ the centered data matrix. The combination of the encoding matrix $E$ with the true class labels $y$ of the MPI database yields the *true* dataset, whereas its combination with the class labels $y_{est}$ by the subjects yields the $subject$ dataset.

To model classification in human subjects we use methods from supervised machine learning. In particular, we consider linear classifiers where classification is done using a SH defined by its normal vector $\vec{w}$ and offset $b$. Furthermore the normal vector $\vec{w}$ of our classifiers can then be written as a linear combination of the input patterns $\vec{x}_i$ with suitable coefficients $\alpha_i$ as $\vec{w} = \sum_i \alpha_i\vec{x}_i$. We define the distance of a pattern to the SH as $\delta(\vec{x}) = \frac{\langle\vec{w}|\vec{x}\rangle+b}{\|\vec{w}\|}$. Note that in our experiments the $\vec{x}_i$ are the PCA coefficients of the images, that is $\vec{x}_i \in \mathbb{R}^{200}$, whereas the images themselves are in $\mathbb{R}^{256^2}$. For the subject dataset we chose the mean values of $\vec{w}$, $b$ and $\vec{w}_\pm$ over all subjects.

## 2.1 Machine Classifiers

The *Support Vector Machine* (SVM, [9, 10]) is a state-of-the-art maximum margin algorithm based on statistical learning theory. SVMs have an intuitive geometrical interpretation: they classify by maximizing the margin separating both classes while minimizing the classification error.

The *Relevance Vector Machine* (RVM, [11]) is a probabilistic Bayesian classifier. It optimises the expansion coefficients of a SV-style decision function using a hyperprior which favours sparse solutions.

Common classifiers in neuroscience, cognitive science and psychology are variants of the *Prototype classifier* (Prot, [12]). Their popularity is due to their simplicity: they classify according to the nearest mean-of-class prototype; in the simplest form all dimensions are weighted equally but variants exist that weight the dimensions inversely proportional the class variance along the dimensions. As we cannot estimate class variance along all 200 dimensions from only 200 stimuli, we chose to implement the simplest Prot with equal weight along all dimensions.

The *Fisher linear discriminant classifier* (FLD, [13]) finds a direction in the dataset which allows best linear separation of the two classes. This direction is then used as the normal vector of the separating hyperplane. In fact, FLD is arguably a more principled whitened variant of the Prot classifier: Its weight vector can be written as $\vec{w} = S_W^{-1}(\vec{\mu}_+ - \vec{\mu}_-)$, where $S_W^{-1}$ is the within class covariance matrix of the two classes, and $\mu_\pm$ are the class means. Consequently, if we disregard the constant offset $b$, we can write the decision function as $\langle \vec{w}|\vec{x}\rangle = \langle S_W^{-1}(\vec{\mu}_+ - \vec{\mu}_-)|\vec{x}\rangle = \langle S_W^{-1/2}(\vec{\mu}_+ - \vec{\mu}_-)|S_W^{-1/2}\vec{x}\rangle$, which is a prototype classifier using the prototypes $\vec{\mu}_\pm$ after whitening the space with $S_W^{-1/2}$.

## 2.2 Decision-Images and Generalised Portraits

We combine the linear preprocessor (PCA) $\bar{X} = EB$ and the linear classifier (SVM, RVM, Prot, FLD) $y(\vec{x}) = \langle \vec{w}|\vec{x}\rangle + b$ to yield a linear classification system: $\vec{y} = \vec{w}^T E^T + \vec{b}$ where $\vec{b} = b\vec{1}$. We define the *decision-image* as the vector $\vec{W}$ effectively used for classification as: $\vec{y} = \vec{W}^T \bar{X}^T + \vec{b}$. We then have $\vec{w}^T E^T = \vec{W}^T \bar{X}^T \Leftrightarrow \vec{w}^T B^{-T} \bar{X}^T = \vec{W}^T \bar{X}^T$ where $B^{-1}$ is the pseudo-inverse of $B$. For the last condition, we obtain a definition of the decision-image $\vec{W} = B^{-1}\vec{w} \in \mathbb{R}^{256^2}$. In the case of PCA where $B^{-1} = B^T$, we simply have $\vec{W} = B^T\vec{w}$.

Figure 1 shows the decision-images $\vec{W}$ for the four classifiers, SVM, RVM, Prot and FLD. The decision-images in the first row are those obtained if the classifiers are trained on the true dataset; those in the second row if trained on the subject dataset, marked on the right hand side of the figure by "true data" and "subj data", respectively. Decision-images are represented by a vector pointing to the positive class and can thus be expected to have male attributes (the negative of it looks female). Both dark and light regions are more important for classification than the grey regions. Inspection of the decision-images is instructive. For the prototype learner, the eye and beard regions are most important. SVM, RVM and FLD have somewhat more "holistic" decision-images. Equally instructive is the comparison of the optimal decision-images of the machine classifiers in row one (0 to 1% classification error for SVM, RVM and FLD) and those trained on the subject labels in row two (the average subject error is 16 % when classifying the faces; the machines attempt to re-create the decision boundaries of the subjects and thus show similar mis-classification errors). The decision-images for the subject dataset are slightly more "face-like" and less holistic than those obtained using the true labels; the eye and mouth regions are more strongly emphasised. This trend is true across all classifiers. This suggest that human subjects base their gender classification strongly on the eye and mouth regions of the face—clearly a sub-optimal strategy as revealed by the more holistic true dataset SVM, RVM and FLD

decision-images.

A decision-image thus represents a way to extract the visual cues and features used by human subjects during visual classification without using *a priori* assumptions or knowledge about the task at hand.

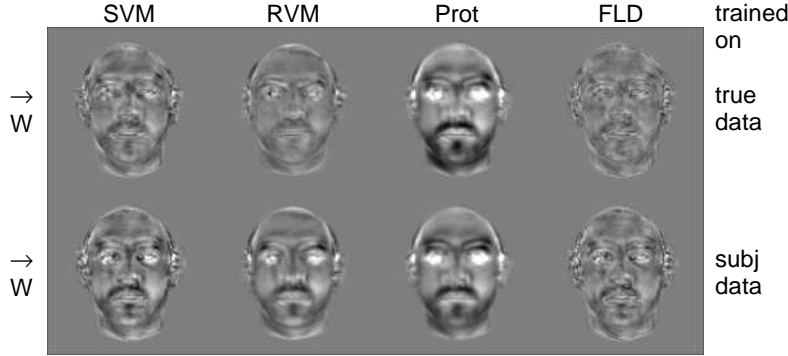

Figure 1: Decision-images $\vec{W}$ for each classifier for both the true and the subject dataset; all images are rescaled to $[0, 1]$ and their means set to $128$ for illustration purposes (different scalers for different images).

We can also define *generalised portraits*[2] $\vec{W}_{\pm}$. The generalised portraits $\vec{W}_{\pm}$ can be seen as "summary" faces in each class reflecting the decision rule of the classifier. They can be viewed as an extension of the concept of a prototype: they are the prototype of the faces the classifier bases its decision on. We note that $\vec{w}$ can be written as: $\vec{w} = \sum_i \alpha_i \vec{x}_i = \sum_{i|\,\text{sign}(\alpha_i)=+1} \alpha_i \vec{x}_i - \sum_{i|\,\text{sign}(\alpha_i)=-1} |\alpha_i| \vec{x}_i$. This allows to define the generalized portraits as $\vec{W}_{\pm}$ which are computed by inverting the PCA transformation on the patterns $\vec{w}_{\pm} = \frac{\sum_{i|\,\text{sign}(\alpha_i)=\pm1} \alpha_i \vec{x}_i}{\sum_{i|\,\text{sign}(\alpha_i)=\pm1} \alpha_i}$. The vector $\vec{w}_{\pm}$ is constrained to be in the convex hull of the respective data in order to yield a "viewable" portrait. The generalised portraits for the SVM, RVM and FLD together with the Prot, where the prototype is the same as the generalised portrait, are shown in figure 2. We also note that $\vec{w}$ can be written as $\vec{w} = \sum_i \alpha_i \vec{x}_i = \sum_{i|\,\text{sign}(\alpha_i)=+1} \alpha_i \vec{x}_i - \sum_{i|\,\text{sign}(\alpha_i)=-1} |\alpha_i| \vec{x}_i$.

The generalised portraits can be associated with the correct class: $\vec{W}_+$ are males whereas $\vec{W}_-$ are females. The SVM and the FLD use patterns close to the SH for classification and hence their decision-images appear androgynous, whereas Prot and RVM tend to use patterns distant from the SH resulting in more female and male generalised portraits. Comparison of the optimal, true, generalised portraits to those based on the subject labels shows that classification has become more difficult: generalised portraits have moved closer to each other in gender space, narrowing the distance between the classes and thereby diminishing the gender typicality of the generalised portraits for all classifiers.

## 3 Human Gender Discrimination along the Decision-Image Axes

The decision-images introduced in section 2.2 are based purely on machine learning, albeit on labels provided by human subjects in the case of the subject dataset. Our previous paper [1] reported that the subjects' responses to the faces—proportion correct, reaction times

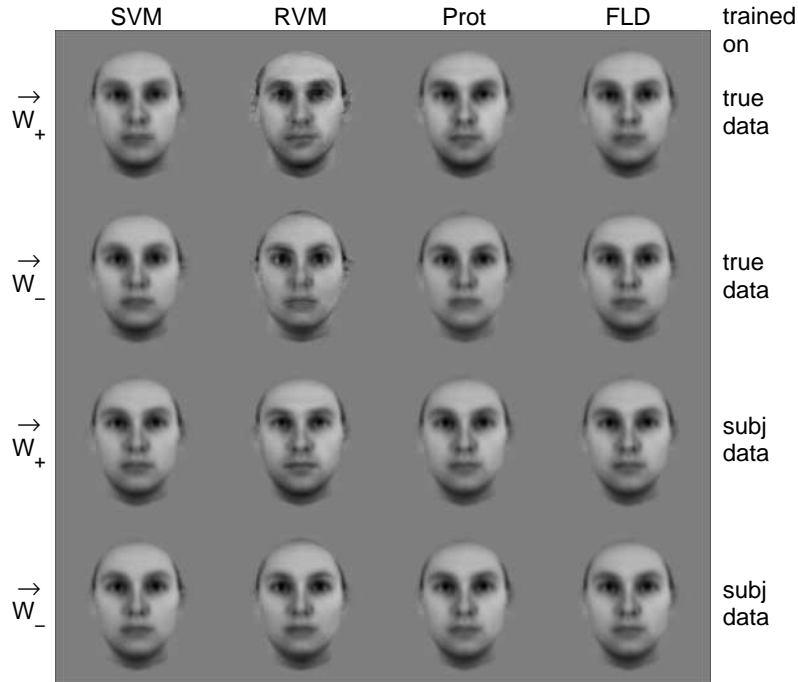

Figure 2: Generalised portraits $\vec{W}_{\pm}$ for each classifier for both the true and the subject dataset; all images are rescaled to $[0, 1]$ and their means set to 128 for illustration purposes (different scalers for different images). [Unfortunately the downsampling (low-pass filtering) of the faces necessary to fit them in the figure makes all the faces somewhat more androgynous than they are viewed at full resolution.]

(RT) and confidence ratings—correlated very well with the distance of the stimuli to their separating hyperplane (SH) for support and relevance vector machines (SVMs, RVMs) but not for simple prototype (Prot) classifier. If these correlations really implied that SVM and RVM capture some crucial aspects of human internal face representation the following prediction must hold: already for small $|\lambda|$ $I_{\text{SVM}}(\lambda)$ and $I_{\text{RVM}}(\lambda)$ should look male/female whereas $|\lambda|$ $I_{\text{Prot}}(\lambda)$ and $I_{\text{FLD}}(\lambda)$ should only be perceptually male/female for larger $|\lambda|$. In other words: the female-to-maleness axis of SVM and RVM should be closely aligned to those of our subjects whereas that is not expected to be the case for FLD and Prot.

### 3.1 Psychophysical Methods

Four observers—one of the authors (FAW) with extensive psychophysical training and three naïve subjects paid for their participation—took part in a standard, spatial (left versus right) two-alternative forced-choice (2AFC) discrimination experiment. Subjects were presented with two faces $I(-\lambda)$ and $I(\lambda)$ and had to indicate which face looked more male. Stimuli were presented against the mean luminance ($50\,\text{cd/m}^2$) of a carefully linearised Clinton Monoray CRT driven by a Cambridge Research Systems VSG 2/5 display controller. Neither male nor female faces changed the mean luminance. Subjects viewed the screen binocularly with their head stabilised by a headrest. The temporal envelope of stimulus presentation was a modified Hanning window (a raised cosine function with rise and fall times of 500 ms and a plateau time of 1000 ms). The probability of the female face being presented on the left was 0.5 on each trial and observers indicated whether they

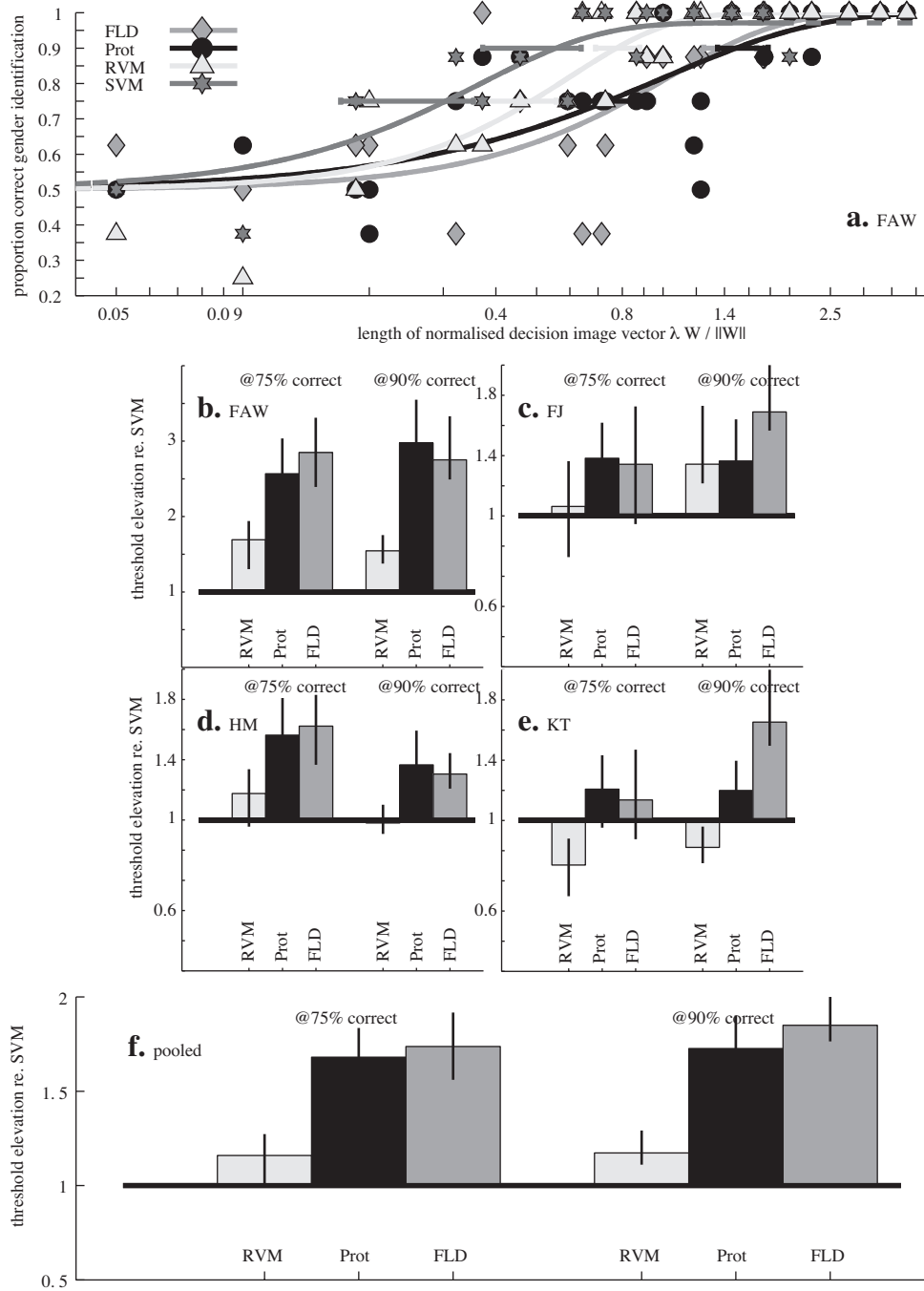

Figure 3: a. Shows raw data and fitted psychometric functions for one observer (FAW). b–e. For each of four observers the threshold elevation for the RVM, Prot and FLD decision-image relative to that of the SVM; results are shown for both 75 and 90% correct together with 68%-CIs. f. Same as in b–e but pooled across observers.

thought the left or right face was female by touching the corresponding location on a Elo TouchSystems touch-screen immediately in front of the display; no feedback was provided.

Trials were run in blocks of 256 in which eight repetitions of eight stimulus levels ($\pm\lambda_1 \ldots \pm \lambda_8$) for each of the four classifiers were randomly intermixed. The naïve subjects required approximately 2000 trials before their performance stabilised; thereafter they did another five to six blocks of 256 trials. All results presented below are based on the trials after training; all training trials were discarded.

## 3.2  Results and Discussion

Figure 3a shows the raw data and fitted psychometric functions for one of the observers. Proportion correct gender identification on the y-axis is plotted against $\lambda$ on the x-axis on semi-logarithmic coordinates. Psychometric functions were fitted using the psignifit toolbox for Matlab which implements the constrained maximum-likelihood method described in [15]. 68%-confidence intervals (CIs), indicated by horizontal lines at 75 and 90-% correct in figure 3a, were estimated by the BCa bootstrap method also implemented in psignifit [16]. The raw data appear noisy because each data point is based on only eight trials. However, none of fitted psychometric functions failed various Monte Carlo based goodness-of-fit tests [15].

To summarise the data we extracted the $\lambda$ required for two performance levels ("thresholds"), 75 and 90% correct, together with their corresponding 68%-CIs. Figure 3b–e shows the thresholds for all four observers normalised by $\lambda_{\mathrm{SVM}}$ (the "threshold elevation" re. SVM). Thus values larger than 1.0 for RVM, Prot and FLD indicate that more of the corresponding decision-images had to be added for the human observers to be able to discriminate females from males. In figure 3f we pool the data across observers as the main trend, poorer performance for Prot and FLD compared to SVM and RVM, is apparent for all four observers. The difference between SVM and RVM is small; going along the direction of both Prot and FLD, however, results in a much "slower" transition from female-to-maleness.

The psychophysical data are very clear: all observers require a larger $\lambda$ for Prot and FLD; the length ratio ranges from 1.2 to nearly 3.0, and averages to around 1.7 across observers. In the pooled data all the differences are statistically significant but even at the individual subject level all differences are significant at the 90% performance level, and five of eight are significant at the 75% performance level. It thus appears that SVM and RVM capture more of the psychological face-space of our human observers than Prot and FLD. From our results we cannot exclude the possibility that some other direction might have yielded even steeper psychometric functions, i.e. faster female-to-maleness transitions, but we can conclude that the decision-images of SVM and RVM are closer to the decision-images used by human subjects than those of Prot and FLD. This is exactly as predicted by the correlations between proportion correct, RTs and confidence ratings versus distance to the hyperplane reported in [1]—high correlations for SVM and RVM, low correlations for Prot.

## 4  Summary and Conclusions

We studied classification and discrimination of human faces both psychophysically as well as using methods from machine learning. The combination of linear preprocessor (PCA) and classifier (SVM, RVM, Prot and FLD) allowed us to visualise the *decision-images* of a classifier corresponding to the vector normal to the SH of the classifier. Decision-images can be used to determine the regions of the stimuli most useful for classification simply by analysing the distribution of light and dark regions in the decision-image. In addition we defined the *generalised portraits* to be the prototypes of all faces used by the classifier to obtain its classification. For the SVM this is the weighted average of all the support

vectors (SVs), for the RVM the weighted average of all the relevance vectors (RVs), and for the Prot it is the prototype itself. The generalised portraits are, like the decision-images, another useful visualisation of the categorisation algorithm of the machine classifier.

However, the central result of our paper is the corroboration of the machine-learning-psychophysics research methodology. In the machine-learning-psychophysics research we substitute a very hard to analyse complex system (the human brain) by a reasonably complex system (learning machine) that is complex enough to capture essentials of our human subjects' behaviour but is nonetheless amenable to close analysis. From the analysis of the machines we then derive predictions for human subjects which we subsequently test psychophysically.

Given the success in predicting the steepness of the female-to-male transition of the $\vec{w}_{\mathrm{SVM}}$ -axis we believe that the decision-image $\vec{W}_{\mathrm{SVM}}$ captures some of the essential characteristics of the human decision algorithm.

**Acknowledgements**   The authors would like to thank Bruce Henning, Frank Jäkel, Ulrike von Luxburg and Christian Wallraven for helpful comments and suggestions. In addition we thank Frank Jäkel for supplying us with the code to run the touch-screen experiment.

## Footnotes

[1]The MPI face database is located at `http://faces.kyb.tuebingen.mpg.de`

[2]This term was introduced by [14] with the idea in mind that when trained on a set of portraits of members of a family, one would obtain a "generalized" portrait which captures the essential features of the family as a superposition of all family members.

# References

[1] A.B.A. Graf and F.A. Wichmann. Insights from machine learning applied to human visual classification. In *Advances in Neural Information Processing Systems 16*. MIT Press, 2004.

[2] M.S. Gray, D.T. Lawrence, B.A. Golomb, and T.S. Sejnowski. A perceptron reveals the face of sex. *Neural Computation*, 7(6):1160–1164, 1995.

[3] P.J.B. Hancock, V. Bruce, and A.M. Burton. A comparison of two computer-based face recognition systems with human perceptions of faces. *Vision Research*, 38:2277–2288, 1998.

[4] A.J. O'Toole, P.J. Phillips, Y. Cheng, B. Ross, and H.A. Wild. Face recognition algorithms as models of human face processing. In *Proceedings of the 4th IEEE International Conference on Automatic Face and Gesture Recognition*, 2000.

[5] B. Moghaddam and M.-H. Yang. Learning gender with support faces. *IEEE Transactions on Pattern Analysis and Machine Intelligence*, 24(5):707–711, 2002.

[6] F. Gosselin and P.G. Schyns. Bubbles: a technique to reveal the use of information in recognition tasks. *Vision Research*, 41:2261–2271, 2001.

[7] A.J. Ahumada Jr. Classification image weights and internal noise level estimation. *Journal of Vision*, 2:121–131, 2002.

[8] M. Turk and A. Pentland. Eigenfaces for recognition. *Journal of Cognitive Neuroscience*, 3(1), 1991.

[9] V.N. Vapnik. *The Nature of Statistical Learning Theory*. Springer, second edition, 2000.

[10] B. Schölkopf and A.J. Smola. *Learning with Kernels*. MIT Press, 2002.

[11] M.E. Tipping. Sparse Bayesian learning and the relevance vector machine. *Journal of Machine Learning Research*, 1:211–214, 2001.

[12] S.K. Reed. Pattern recognition and categorization. *Cognitive Psychology*, 3:382–407, 1972.

[13] R. A. Fisher. The use of multiple measurements in taxonomic problems. *Annals of Eugenics*, 7(2):179–188, 1936.

[14] V. Vapnik and A. Lerner. Pattern recognition using generalized portrait method. *Automation and Remote Control*, 24:774–780, 1963.

[15] F.A. Wichmann and N.J. Hill. The psychometric function: I. fitting, sampling and goodness-of-fit. *Perception and Psychophysics*, 63(8):1293–1313, 2001.

[16] F.A. Wichmann and N.J. Hill. The psychometric function: II. bootstrap-based confidence intervals and sampling. *Perception and Psychophysics*, 63(8):1314–1329, 2001.
